# Multiple Kernel Learning and the SMO Algorithm

**S. V. N. Vishwanathan, Zhaonan Sun, Nawanol Theera-Ampornpunt**
Purdue University
vishy@stat.purdue.edu, sunz@stat.purdue.edu, ntheeraa@cs.purdue.edu

**Manik Varma**
Microsoft Research India
manik@microsoft.com

## Abstract

Our objective is to train $p$-norm Multiple Kernel Learning (MKL) and, more generally, linear MKL regularised by the Bregman divergence, using the Sequential Minimal Optimization (SMO) algorithm. The SMO algorithm is simple, easy to implement and adapt, and efficiently scales to large problems. As a result, it has gained widespread acceptance and SVMs are routinely trained using SMO in diverse real world applications. Training using SMO has been a long standing goal in MKL for the very same reasons. Unfortunately, the standard MKL dual is not differentiable, and therefore can not be optimised using SMO style co-ordinate ascent. In this paper, we demonstrate that linear MKL regularised with the $p$-norm squared, or with certain Bregman divergences, can indeed be trained using SMO. The resulting algorithm retains both simplicity and efficiency and is significantly faster than state-of-the-art specialised $p$-norm MKL solvers. We show that we can train on a hundred thousand kernels in approximately seven minutes and on fifty thousand points in less than half an hour on a single core.

## 1 Introduction

Research on Multiple Kernel Learning (MKL) needs to follow a two pronged approach. It is important to explore formulations which lead to improvements in prediction accuracy. Recent trends indicate that performance gains can be achieved by non-linear kernel combinations [7,18,21], learning over large kernel spaces [2] and by using general, or non-sparse, regularisation [6, 7, 12, 18]. Simultaneously, efficient optimisation techniques need to be developed to scale MKL out of the lab and into the real world. Such algorithms can help in investigating new application areas and different facets of the MKL problem including dealing with a very large number of kernels and data points.

Optimisation using decompositional algorithms such as Sequential Minimal Optimization (SMO) [15] has been a long standing goal in MKL [3] as the algorithms are simple, easy to implement and efficiently scale to large problems. The hope is that they might do for MKL what SMO did for SVMs – allow people to play with MKL on their laptops, modify and adapt it for diverse real world applications and explore large scale settings in terms of number of kernels and data points.

Unfortunately, the standard MKL formulation, which learns a linear combination of base kernels subject to $l_1$ regularisation, leads to a dual which is not differentiable. SMO can not be applied as a result and [3] had to resort to expensive Moreau-Yosida regularisation to smooth the dual. State-of-the-art algorithms today overcome this limitation by solving an intermediate saddle point problem rather than the dual itself [12, 16].

Our focus, in this paper, is on training $p$-norm MKL, with $p > 1$, using the SMO algorithm. More generally, we prove that linear MKL regularised by certain Bregman divergences, can also be trained

using SMO. We shift the emphasis firmly back towards solving the dual in such cases. The $l_p$-MKL dual is shown to be differentiable and thereby amenable to co-ordinate ascent. Placing the $p$-norm squared regulariser in the objective lets us efficiently solve the core reduced two variable optimisation problem analytically in some cases and algorithmically in others. Using results from [4, 9], we can compute the $l_p$-MKL Hessian, which brings into play second order variable selection methods which tremendously speed up the rate of convergence [8]. The standard decompositional method proof of convergence [14] to the global optimum holds with minor modifications.

The resulting optimisation algorithm, which we call SMO-MKL, is straight forward to implement and efficient. We demonstrate that SMO-MKL can be significantly faster than the state-of-the-art specialised $p$-norm solvers [12]. We empirically show that the SMO-MKL algorithm is robust with the desirable property that it is not greatly affected within large operating ranges of $p$. This implies that our algorithm is well suited for learning both sparse, and non-sparse, kernel combinations. Furthermore, SMO-MKL scales well to large problems. We show that we can efficiently combine a hundred thousand kernels in approximately seven minutes or train on fifty thousand points in less than half an hour using a single core on standard hardware where other solvers fail to produce results. The SMO-MKL code can be downloaded from [20].

## 2   Related Work

Recent trends indicate that there are three promising directions of research for obtaining performance improvements using MKL. The first involves learning non-linear kernel combinations. A framework for learning general non-linear kernel combinations subject to general regularisation was presented in [18]. It was demonstrated that, for feature selection, the non-linear GMKL formulation could perform significantly better not only as compared to linear MKL but also state-of-the-art wrapper methods and filter methods with averaging. Very significant performance gains in terms of pure classification accuracy were reported in [21] by learning a different kernel combination per data point or cluster. Again, the results were better not only as compared to linear MKL but also baselines such as averaging. Similar trends were observed for regression while learning polynomial kernel combinations [7]. Other promising directions which have resulted in performance gains are sticking to standard MKL but combining an exponentially large number of kernels [2] and linear MKL with $p$-norm regularisers [6, 12]. Thus MKL based methods are beginning to define the state-of-the-art for very competitive applications, such as object recognition on the Caltech 101 database [21] and object detection on the PASCAL VOC 2009 challenge [19].

In terms of optimisation, initial work on MKL leveraged general purpose SDP and QCQP solvers [13]. The SMO+M.-Y. regularisation method of [3] was one of the first techniques that could efficiently tackle medium scale problems. This was superseded by the SILP technique of [17] which could, very impressively, train on a million point problem with twenty kernels using parallelism. Unfortunately, the method did not scale well with the number of kernels. In response, many two-stage wrapper techniques came up [2, 10, 12, 16, 18] which could be significantly faster when the number of training points was reasonable but the number of kernels large. SMO could indirectly be used in some of these cases to solve the inner SVM optimisation. The primary disadvantage of these techniques was that they solved the inner SVM to optimality. In fact, the solution needed to be of high enough precision so that the kernel weight gradient computation was accurate and the algorithm converged. In addition, Armijo rule based step size selection was also very expensive and could involve tens of inner SVM evaluations in a single line search. This was particularly expensive since the kernel cache would be invalidated from one SVM evaluation to the next. The one big advantage of such two-stage methods for $l_1$-MKL was that they could quickly identify, and discard, the kernels with zero weights and thus scaled well with the number of kernels. Most recently, [12] have come up with specialised $p$-norm solvers which make substantial gains by not solving the inner SVM to optimality and working with a small active set to better utilise the kernel cache.

## 3   The $l_p$-MKL Formulation

The objective in MKL is to jointly learn kernel and SVM parameters from training data $\{(\mathbf{x}_i, y_i)\}$. Given a set of base kernels $\{K_k\}$ and corresponding feature maps $\{\phi_k\}$, linear MKL aims to learn a linear combination of the base kernels as $K = \sum_k d_k K_k$. If the kernel weights are restricted to

be non-negative, then the MKL task corresponds to learning a standard SVM in the feature space formed by concatenating the vectors $\sqrt{d_k}\boldsymbol{\phi}_k$. The primal can therefore be formulated as

$$\min_{\mathbf{w},b,\boldsymbol{\xi}\geq\mathbf{0},\mathbf{d}\geq\mathbf{0}} \tfrac{1}{2}\sum_k \mathbf{w}_k^t\mathbf{w}_k + C\sum_i \xi_i + \frac{\lambda}{2}(\sum_k d_k^p)^{\frac{2}{p}} \quad \text{s. t. } y_i(\sum_k \sqrt{d_k}\mathbf{w}_k^t\boldsymbol{\phi}_k(\mathbf{x}_i)+b) \geq 1-\xi_i \quad (1)$$

The regularisation on the kernel weights is necessary to prevent them from shooting off to infinity. Which regulariser one uses depends on the task at hand. In this Section, we limit ourselves to the $p$-norm squared regulariser with $p > 1$. If it is felt that certain kernels are noisy and should be discarded then a sparse solution can be obtained by letting $p$ tend to unity from above. Alternatively, if the application demands dense solutions, then larger values of $p$ should be selected. Note that the primal above can be made convex by substituting $\mathbf{w}_k$ for $\sqrt{d_k}\mathbf{w}_k$ to get

$$\min_{\mathbf{w},b,\boldsymbol{\xi}\geq\mathbf{0},\mathbf{d}\geq\mathbf{0}} \tfrac{1}{2}\sum_k \mathbf{w}_k^t\mathbf{w}_k/d_k + C\sum_i \xi_i + \frac{\lambda}{2}(\sum_k d_k^p)^{\frac{2}{p}} \quad \text{s. t. } y_i(\sum_k \mathbf{w}_k^t\boldsymbol{\phi}_k(\mathbf{x}_i)+b) \geq 1-\xi_i \quad (2)$$

We first derive an intermediate saddle point optimisation problem obtained by minimising only $\mathbf{w}$, $b$ and $\boldsymbol{\xi}$. The Lagrangian is

$$L = \tfrac{1}{2}\sum_k \mathbf{w}_k^t\mathbf{w}_k/d_k + \sum_i(C-\beta_i)\xi_i + \frac{\lambda}{2}(\sum_k d_k^p)^{\frac{2}{p}} - \sum_i \alpha_i[y_i(\sum_k \mathbf{w}_k^t\boldsymbol{\phi}_k(\mathbf{x}_i)+b)-1+\xi_i] \quad (3)$$

Differentiating with respect to $\mathbf{w}$, $b$ and $\boldsymbol{\xi}$ to get the optimality conditions and substituting back results in the following intermediate saddle point problem

$$\min_{\mathbf{d}\geq\mathbf{0}}\max_{\boldsymbol{\alpha}\in\mathcal{A}} \mathbf{1}^t\boldsymbol{\alpha} - \tfrac{1}{2}\sum_k d_k\boldsymbol{\alpha}^t H_k\boldsymbol{\alpha} + \frac{\lambda}{2}(\sum_k d_k^p)^{\frac{2}{p}} \quad (4)$$

where $\mathcal{A} = \{\boldsymbol{\alpha}|\mathbf{0} \leq \boldsymbol{\alpha} \leq C\mathbf{1}, \mathbf{1}^t Y\boldsymbol{\alpha} = 0\}$, $H_k = YK_kY$ and $Y$ is a diagonal matrix with the labels on the diagonal. Note that most MKL methods end up optimising either this, or a very similar, saddle point problem. To now eliminate $\mathbf{d}$ we again form the Lagrangian

$$L = \mathbf{1}^t\boldsymbol{\alpha} - \tfrac{1}{2}\sum_k d_k\boldsymbol{\alpha}^t H_k\boldsymbol{\alpha} + \frac{\lambda}{2}(\sum_k d_k^p)^{\frac{2}{p}} - \sum_k \gamma_k d_k \quad (5)$$

$$\frac{\partial L}{\partial d_k} = 0 \Rightarrow \lambda(\sum_k d_k^p)^{\frac{2}{p}-1}d_k^{p-1} = \gamma_k + \tfrac{1}{2}\boldsymbol{\alpha}^t H_k\boldsymbol{\alpha} \quad (6)$$

$$\Rightarrow \lambda(\sum_k d_k^p)^{\frac{2}{p}} = \sum_k d_k(\gamma_k + \tfrac{1}{2}\boldsymbol{\alpha}^t H_k\boldsymbol{\alpha}) \quad (7)$$

$$\Rightarrow L = \mathbf{1}^t\boldsymbol{\alpha} - \frac{\lambda}{2}(\sum_k d_k^p)^{\frac{2}{p}} = \mathbf{1}^t\boldsymbol{\alpha} - \frac{1}{2\lambda}(\sum_k(\gamma_k + \tfrac{1}{2}\boldsymbol{\alpha}^t H_k\boldsymbol{\alpha})^q)^{\frac{2}{q}} \quad (8)$$

where $\frac{1}{p} + \frac{1}{q} = 1$. Since $H_k$ is positive semi-definite, $\boldsymbol{\alpha}^t H_k\boldsymbol{\alpha} \geq 0$ and since $\gamma_k \geq 0$ it is clear that the optimal value of $\gamma_k$ is zero. Our $l_p$-MKL dual therefore becomes

$$D \equiv \max_{\boldsymbol{\alpha}\in\mathcal{A}} \mathbf{1}^t\boldsymbol{\alpha} - \frac{1}{8\lambda}(\sum_k(\boldsymbol{\alpha}^t H_k\boldsymbol{\alpha})^q)^{\frac{2}{q}} \quad (9)$$

and the kernel weights can be recovered from the dual variables as

$$d_k = \frac{1}{2\lambda}\left(\sum_k(\boldsymbol{\alpha}^t H_k\boldsymbol{\alpha})^q\right)^{\frac{1}{q}-\frac{1}{p}}(\boldsymbol{\alpha}^t H_k\boldsymbol{\alpha})^{\frac{q}{p}} \quad (10)$$

Note that our dual objective, unlike the objective in [3], is differentiable with respect to $\boldsymbol{\alpha}$. The SMO algorithm can therefore be brought to bear where two variables are selected and optimised using gradient or Newton methods and the process repeated until convergence.

Also note that it has sometimes been observed that $l_2$ regularisation can provide better results than $l_1$ [6, 7, 12, 18]. For this special case, when $p = q = 2$, the reduced two variable problem can be solved analytically. This was one of the primary motivations for choosing the $p$-norm squared regulariser and placing it in the primal objective (the other was to be consistent with other $p$-norm formulations [9, 11]). Had we included the regulariser as a primal constraint then the dual would have the $q$-norm rather than the $q$-norm squared. Our dual would then be near identical to Eq. (9) in [12]. However, it would then no longer have been possible to solve the two variable reduced problem analytically for the 2-norm special case.

## 4  SMO-MKL Optimisation

We now develop the SMO-MKL algorithm for optimising the $l_p$ MKL dual. The algorithm has three main components: (a) reduced variable optimisation; (b) working set selection and (c) stopping criterion and kernel caching. We build the SMO-MKL algorithm around the LibSVM code base [5].

### 4.1  The Reduced Variable Optimisation

The SMO algorithm works by repeatedly choosing two variables (assumed to be $\alpha_1$ and $\alpha_2$ without loss of generality in this Subsection) and optimising them while holding all other variables constant. If $\alpha_1 \leftarrow \alpha_1 + \Delta$ and $\alpha_2 \leftarrow \alpha_2 + s\Delta$, the dual simplifies to

$$\Delta^* = \operatorname*{argmax}_{L \leq \Delta \leq U} \ (1+s)\Delta - \frac{1}{8\lambda}(\sum_k (a_k\Delta^2 + 2b_k\Delta + c_k)^q)^{\frac{2}{q}} \tag{11}$$

where $s = -y_1 y_2$, $L = (s == +1)$ ? $\max(-\alpha_1, -\alpha_2)$ : $\max(-\alpha_1, \alpha_2 - C)$, $U = (s == +1)$ ? $\min(C - \alpha_1, C - \alpha_2)$ : $\min(C - \alpha_1, \alpha_2)$, $a_k = H_{11k} + H_{22k} + 2sH_{12k}$, $b_k = \boldsymbol{\alpha}^t(H_{:1k} + sH_{:2k})$ and $c_k = \boldsymbol{\alpha}^t H_k \boldsymbol{\alpha}$. Unlike as in SMO, $\Delta^*$ can not be found analytically for arbitrary $p$. Nevertheless, since this is a simple one dimensional concave optimisation problem, we can efficiently find the global optimum using a variety of methods. We tried bisection search and Brent's algorithm but the Newton-Raphson method worked best – partly because the one dimensional Hessian was already available from the working set selection step.

### 4.2  Working Set Selection

The choice of which two variables to select for optimisation can have a big impact on training time. Very simple strategies, such as random sampling, can have very little cost per iteration but need many iterations to converge. First and second order working set selection techniques are more expensive per iteration but converge in far fewer iterations.

We implement the greedy second order working set selection strategy of [8]. We do not give the variable selection equations due to lack of space but refer the interested reader to the WSS2 method of [8] and our source code [20]. The critical thing is that the selection of the first (second) variable involves computing the gradient (Hessian) of the dual. These are readily derived to be

$$\nabla_{\boldsymbol{\alpha}} D = \mathbf{1} - \sum_k d_k H_k \boldsymbol{\alpha} = \mathbf{1} - H\boldsymbol{\alpha} \tag{12}$$

$$\nabla_{\boldsymbol{\alpha}}^2 D = -H - \frac{1}{\lambda}\sum_k \nabla_{\theta_k} f^{-1}(\boldsymbol{\theta})(H_k\boldsymbol{\alpha})(H_k\boldsymbol{\alpha})^t \tag{13}$$

where $\nabla_{\theta_k} f^{-1}(\boldsymbol{\theta}) = (2-q)\boldsymbol{\theta}_q^{2-2q}\theta_k^{2q-2} + (q-1)\boldsymbol{\theta}_q^{2-q}\theta_k^{q-2}$ and $\theta_k = \frac{1}{2\lambda}\boldsymbol{\alpha}^t H_k \boldsymbol{\alpha}$ (14)

where $D$ has been overloaded to now refer to the dual objective. Rather than compute the gradient $\nabla_{\boldsymbol{\alpha}} D$ repeatedly, we speed up variable selection by caching, separately for each kernel, $H_k\boldsymbol{\alpha}$. The cache needs to be updated every time we change $\boldsymbol{\alpha}$ in the reduced variable optimisation. However, since only two variables are changed, $H_k\boldsymbol{\alpha}$ can be updated by summing along just two columns of the kernel matrix. This involves only $O(M)$ work in all, where $M$ is the number of kernels, since the column sums can be pre-computed for each kernel. The Hessian is too expensive to cache and is recomputed on demand.

### 4.3  Stopping Criterion and Kernel Caching

We terminate the SMO-MKL algorithm when the duality gap falls below a pre-specified threshold. Kernel caching strategies can have a big impact on performance since kernel computations can dominate everything else in some cases. While a few different kernel caching techniques have been explored for SVMs, we stick to the standard one used in LibSVM [5]. A Least Recently Used (LRU) cache is implemented as a circular queue. Each element in the queue is a pointer to a recently accessed (common) row of each of the individual kernel matrices.

# 5 Special Cases and Extensions

We briefly discuss a few special cases and extensions which impact our SMO-MKL optimisation.

## 5.1 2-Norm MKL

As we noted earlier, 2-norm MKL has sometimes been found to outperform MKL trained with $l_1$ regularisation [6, 7, 12, 18]. For this special case, when $p = q = 2$, our dual and reduced variable optimisation problems simplify to polynomials of degree four

$$D_2 \equiv \max_{\boldsymbol{\alpha} \in \mathcal{A}} \mathbf{1}^t \boldsymbol{\alpha} - \frac{1}{8\lambda} \sum_k (\boldsymbol{\alpha}^t H_k \boldsymbol{\alpha})^2 \tag{15}$$

$$\Delta^* = \operatorname*{argmax}_{L \leq \Delta \leq U} (1+s)\Delta - \frac{1}{8\lambda} \sum_k (a_k \Delta^2 + 2b_k \Delta + c_k)^2 \tag{16}$$

Just as in standard SMO, $\Delta^*$ can now be found analytically by using the expressions for the roots of a cubic. This makes our SMO-MKL algorithm particularly efficient for $p = 2$ and our code defaults to the analytic solver for this special case.

## 5.2 The Bregman Divergence as a Regulariser

The Bregman divergence generalises the squared $p$-norm. It is not a metric as it is not symmetric and does not obey the triangle inequality. In this Subsection, we demonstrate that our MKL formulation can also incorporate the Bregman divergence as a regulariser.

Let $F$ be any differentiable, strictly convex function and $f = \nabla F$ represent its gradient. The Bregman divergence generated by $F$ is given by $r_F(\mathbf{d}) = F(\mathbf{d}) - F(\mathbf{d}_0) - (\mathbf{d} - \mathbf{d}_0)^t f(\mathbf{d}_0)$. Note that $\nabla r_F(\mathbf{d}) = f(\mathbf{d}) - f(\mathbf{d}_0)$. Incorporating the Bregman divergence as a regulariser in our primal objective leads to the following intermediate saddle point problem and Lagrangian

$$I_B \equiv \min_{\mathbf{d} \geq 0} \max_{\boldsymbol{\alpha} \in \mathcal{A}} \mathbf{1}^t \boldsymbol{\alpha} - \tfrac{1}{2} \sum_k d_k \boldsymbol{\alpha}^t H_k \boldsymbol{\alpha} + \lambda r_F(\mathbf{d}) \tag{17}$$

$$L_B = \mathbf{1}^t \boldsymbol{\alpha} - \sum_k d_k (\gamma_k + \tfrac{1}{2} \boldsymbol{\alpha}^t H_k \boldsymbol{\alpha}) + \lambda r_F(\mathbf{d}) \tag{18}$$

$$\nabla_{\mathbf{d}} L_B = 0 \Rightarrow f(\mathbf{d}) - f(\mathbf{d}_0) = g(\boldsymbol{\alpha}, \gamma)/\lambda \tag{19}$$

$$\Rightarrow \mathbf{d} = f^{-1}\left(f(\mathbf{d}_0) + g(\boldsymbol{\alpha}, \gamma)/\lambda\right) = f^{-1}(\boldsymbol{\theta}(\boldsymbol{\alpha}, \gamma)) \tag{20}$$

where $g$ is a vector with entries $g_k(\boldsymbol{\alpha}, \gamma) = \gamma_k + \tfrac{1}{2} \boldsymbol{\alpha}^t H_k \boldsymbol{\alpha}$ and $\boldsymbol{\theta}(\boldsymbol{\alpha}, \gamma) = f(\mathbf{d}_0) + g(\boldsymbol{\alpha}, \gamma)/\lambda$. Substituting back in the Lagrangian and discarding terms dependent on just $\mathbf{d}_0$ results in the dual

$$D_R \equiv \max_{\boldsymbol{\alpha} \in \mathcal{A}, \boldsymbol{\gamma} \geq 0} \mathbf{1}^t \boldsymbol{\alpha} + \lambda(F(f^{-1}(\boldsymbol{\theta})) - \boldsymbol{\theta}^t f^{-1}(\boldsymbol{\theta})) \tag{21}$$

In many cases the optimal value of $\gamma$ will turn out to be zero and the optimisation can efficiently be carried out over $\boldsymbol{\alpha}$ using our SMO-MKL algorithm.

**Generalised KL Divergence** To take a concrete example, different from the $p$-norm squared used thus far, we investigate the use of the generalised KL divergence as a regulariser. Choosing $F(\mathbf{d}) = \sum_k d_k(\log(d_k) - 1)$ leads to the generalised KL divergence between $\mathbf{d}$ and $\mathbf{d}_0$

$$r_{KL}(\mathbf{d}) = \sum_k d_k \log(d_k/d_k^0) - \sum_k d_k + \sum_k d_k^0 \tag{22}$$

Plugging in $r_{KL}$ in $I_B$ and following the steps above leads to the following dual problem

$$\max_{\boldsymbol{\alpha} \in \mathcal{A}} \mathbf{1}^t \boldsymbol{\alpha} - \lambda \sum_k d_k^0 e^{\frac{1}{2\lambda} \boldsymbol{\alpha}^t H_k \boldsymbol{\alpha}} \tag{23}$$

which can be optimised straight forwardly using our SMO-MKL algorithm once we plug in the gradient and hessian information. However, discussing this further would take us too far out of the scope of this paper. We therefore stay focused on $l_p$-MKL for the remainder of this paper.

## 5.3 Regression and Other Loss Functions

While we have discussed MKL based classification so far we can easily adapt our formulation to handle other convex loss functions such as regression, novelty detection, *etc*. We demonstrate this for the $\epsilon$-insensitive loss function for regression. The primal, intermediate saddle point and final dual problems are given by

$$P_R \equiv \min_{\mathbf{w},b,\boldsymbol{\xi}^{\pm}\geq\mathbf{0},\mathbf{d}\geq\mathbf{0}} \frac{1}{2}\sum_k \mathbf{w}_k^t\mathbf{w}_k/d_k + C\sum_i(\xi_i^+ + \xi_i^-) + \frac{\lambda}{2}(\sum_k d_k^p)^{\frac{2}{p}} \qquad (24)$$

$$\text{such that} \quad \pm(\sum_k \mathbf{w}_k^t\phi_k(\mathbf{x}_i) + b - y_i) \leq \epsilon + \xi_i^{\pm} \qquad (25)$$

$$I_R \equiv \min_{\mathbf{d}\geq\mathbf{0}} \max_{\leq|\boldsymbol{\alpha}|\leq C\mathbf{1},\ \mathbf{1}^t\boldsymbol{\alpha}=0} \mathbf{1}^t(Y\boldsymbol{\alpha} - \epsilon|\boldsymbol{\alpha}|) - \frac{1}{2}\sum_k d_k\boldsymbol{\alpha}^t K_k\boldsymbol{\alpha} + \frac{\lambda}{2}(\sum_k d_k^p)^{\frac{2}{p}} \qquad (26)$$

$$D_R \equiv \max_{\mathbf{0}\leq|\boldsymbol{\alpha}|\leq C\mathbf{1},\ \mathbf{1}^t\boldsymbol{\alpha}=0} \mathbf{1}^t(Y\boldsymbol{\alpha} - \epsilon|\boldsymbol{\alpha}|) - \frac{1}{8\lambda}(\sum_k(\boldsymbol{\alpha}^t K_k\boldsymbol{\alpha})^q)^{\frac{2}{q}} \qquad (27)$$

SMO has a slightly harder time optimising $D_R$ due to the $|\boldsymbol{\alpha}|$ term which, though in itself not differentiable, can be gotten around by substituting $\boldsymbol{\alpha} = \boldsymbol{\alpha}^+ - \boldsymbol{\alpha}^-$ at the cost of doubling the number of dual variables.

## 6 Experiments

In this Section, we empirically compare the performance of our proposed SMO-MKL algorithm against the specialised $l_p$-MKL solver of [12] which is referred to as Shogun. Code, scripts and parameter settings were helpfully provided by the authors and we ensure that our stopping criteria are compatible. All experiments are carried out on a single core of an AMD 2380 2.5 GHz processor with 32 Gb RAM. Our focus in these experiments is purely on training time and speed of optimisation as the prediction accuracy improvements of $l_p$-MKL have already been documented [12].

We carry out two sets of experiments. The first, on small scale UCI data sets, are carried out using pre-computed kernels. This performs a direct comparison of the algorithmic components of SMO-MKL and Shogun. We also carry out a few large scale experiments with kernels computed on the fly. This experiment compares the two methods in totality. In this case, kernel caching can have an effect, but not a significant one as the two methods have very similar caching strategies.

For each UCI data set we generated kernels as recommended in [16]. We generated RBF kernels with ten bandwidths for each individual dimension of the feature vector as well as the full feature vector itself. Similarly, we also generated polynomial kernels of degrees 1, 2 and 3. All kernels matrices were pre-computed and normalised to have unit trace. We set $C = 100$ as it gives us a reasonable accuracy on the test set. Note that for some value of $\lambda$, SMO-MKL and Shogun will converge to exactly the same solution [12]. Since this value is not known *a priori* we arbitrarily set $\lambda = 1$.

Training times on the UCI data sets are presented in Table 1. Means and standard deviations are reported for five fold cross-validation. As can be seen, SMO-MKL is significantly faster than Shogun at converging to similar solutions and obtaining similar test accuracies. In many cases, SMO-MKL is more than four times as fast and in some case more than ten or twenty times as fast. Note that our test classification accuracy on Liver is a lot lower than Shogun's. This is due to the arbitrary choice of $\lambda$. We can vary our $\lambda$ on Liver to recover the same accuracy and solution as Shogun with a further decrease in our training time.

Another very positive thing is that SMO-MKL appears to be relatively stable across a large operating range of $p$. The code is, in most of the cases as expected, fastest when $p = 2$ and gets slower as one increases or decreases $p$. Interestingly though, the algorithm doesn't appear to be significantly slower for other values of $p$. Therefore, it is hoped that SMO-MKL can be used to learn sparse kernel combinations as well as non-sparse ones.

Moving on to the large scale experiments with kernels computed on the fly, we first tried combining a hundred thousand RBF kernels on the Sonar data set with 208 points and 59 dimensional features.

Table 1: Training times on UCI data sets with $N$ training points, $D$ dimensional features, $M$ kernels and $T$ test points. Mean and standard deviations are reported for 5-fold cross validation.

(a) Australian: $N$=552, $T$=138, $D$=13, $M$=195.

| $p$ | Training Time (s) | | Test Accuracy (%) | | # Kernels Selected | |
|---|---|---|---|---|---|---|
| | SMO-MKL | Shogun | SMO-MKL | Shogun | SMO-MKL | Shogun |
| 1.10 | $4.89 \pm 0.31$ | $58.52 \pm 16.49$ | $85.22 \pm 2.96$ | $85.22 \pm 2.81$ | $26.4 \pm 0.8$ | $137.2 \pm 53.8$ |
| 1.33 | $4.16 \pm 0.16$ | $33.58 \pm 2.58$ | $85.36 \pm 3.79$ | $85.07 \pm 2.85$ | $40.8 \pm 1.3$ | $62.4 \pm 4.7$ |
| 1.66 | $4.31 \pm 0.19$ | $31.89 \pm 1.25$ | $85.65 \pm 3.73$ | $85.07 \pm 2.85$ | $72.2 \pm 4.8$ | $100.2 \pm 3.7$ |
| 2.00 | $4.27 \pm 0.10$ | $27.08 \pm 7.18$ | $85.80 \pm 3.74$ | $85.22 \pm 2.99$ | $126.4 \pm 4.3$ | $134.4 \pm 5.6$ |
| 2.33 | $4.88 \pm 0.18$ | $24.92 \pm 6.46$ | $85.80 \pm 3.74$ | $85.07 \pm 2.85$ | $162.8 \pm 3.6$ | $177.8 \pm 8.3$ |
| 2.66 | $5.19 \pm 0.05$ | $26.90 \pm 2.05$ | $85.80 \pm 3.68$ | $85.22 \pm 2.85$ | $188.2 \pm 4.7$ | $188.8 \pm 5.1$ |
| 3.00 | $5.48 \pm 0.21$ | $27.06 \pm 2.20$ | $85.51 \pm 3.69$ | $85.22 \pm 2.85$ | $192.0 \pm 2.6$ | $194.4 \pm 1.2$ |

(b) Ionosphere: $N$=280, $T$=71, $D$=33, $M$=442.

| $p$ | Training Time (s) | | Test Accuracy (%) | | # Kernels Selected | |
|---|---|---|---|---|---|---|
| | SMO-MKL | Shogun | SMO-MKL | Shogun | SMO-MKL | Shogun |
| 1.10 | $2.85 \pm 0.16$ | $19.82 \pm 4.02$ | $92.60 \pm 1.35$ | $92.03 \pm 1.68$ | $50.0 \pm 2.7$ | $125.2 \pm 7.3$ |
| 1.33 | $2.78 \pm 1.18$ | $8.49 \pm 0.61$ | $92.03 \pm 1.42$ | $92.60 \pm 1.86$ | $120.8 \pm 6.0$ | $217.0 \pm 23.4$ |
| 1.66 | $2.42 \pm 0.28$ | $10.49 \pm 2.27$ | $91.74 \pm 2.08$ | $91.74 \pm 1.37$ | $200.8 \pm 4.4$ | $291.4 \pm 33.0$ |
| 2.00 | $2.16 \pm 0.16$ | $13.99 \pm 4.68$ | $92.03 \pm 1.68$ | $91.17 \pm 2.45$ | $328.0 \pm 6.6$ | $364.2 \pm 15.4$ |
| 2.33 | $2.35 \pm 0.25$ | $24.90 \pm 9.43$ | $92.03 \pm 1.68$ | $91.74 \pm 2.08$ | $413.6 \pm 5.6$ | $412.2 \pm 6.6$ |
| 2.66 | $2.50 \pm 0.32$ | $33.05 \pm 3.66$ | $92.03 \pm 1.68$ | $92.03 \pm 1.68$ | $430.6 \pm 4.6$ | $436.6 \pm 4.3$ |
| 3.00 | $3.03 \pm 0.99$ | $36.23 \pm 3.62$ | $92.31 \pm 1.41$ | $91.75 \pm 2.05$ | $434.4 \pm 4.8$ | $442.0 \pm 0.0$ |

(c) Liver: $N$=276, $T$=69, $D$=5, $M$=91.

| $p$ | Training Time (s) | | Test Accuracy (%) | | # Kernels Selected | |
|---|---|---|---|---|---|---|
| | SMO-MKL | Shogun | SMO-MKL | Shogun | SMO-MKL | Shogun |
| 1.10 | $0.53 \pm 0.03$ | $2.15 \pm 0.12$ | $62.90 \pm 9.81$ | $66.67 \pm 9.91$ | $9.40 \pm 1.02$ | $39.40 \pm 1.50$ |
| 1.33 | $0.54 \pm 0.03$ | $0.92 \pm 0.05$ | $66.09 \pm 8.48$ | $71.59 \pm 8.92$ | $24.40 \pm 2.06$ | $43.60 \pm 2.42$ |
| 1.66 | $0.56 \pm 0.04$ | $1.14 \pm 0.23$ | $66.96 \pm 7.53$ | $70.72 \pm 9.28$ | $44.20 \pm 2.23$ | $57.00 \pm 3.29$ |
| 2.00 | $0.54 \pm 0.04$ | $1.72 \pm 0.57$ | $66.96 \pm 7.06$ | $72.17 \pm 6.94$ | $71.00 \pm 5.29$ | $78.00 \pm 2.28$ |
| 2.33 | $0.63 \pm 0.03$ | $2.35 \pm 0.36$ | $66.38 \pm 7.36$ | $73.33 \pm 6.71$ | $82.40 \pm 2.42$ | $88.20 \pm 1.72$ |
| 2.66 | $0.65 \pm 0.02$ | $2.53 \pm 0.44$ | $65.22 \pm 6.80$ | $72.75 \pm 7.96$ | $83.20 \pm 2.32$ | $90.80 \pm 0.40$ |
| 3.00 | $0.67 \pm 0.03$ | $3.40 \pm 0.55$ | $65.22 \pm 6.74$ | $73.91 \pm 7.28$ | $85.20 \pm 3.37$ | $91.00 \pm 0.00$ |

(d) Sonar: $N$=166, $T$=42, $D$=59, $M$=793.

| $p$ | Training Time (s) | | Test Accuracy (%) | | # Kernels Selected | |
|---|---|---|---|---|---|---|
| | SMO-MKL | Shogun | SMO-MKL | Shogun | SMO-MKL | Shogun |
| 1.10 | $4.95 \pm 0.29$ | $47.19 \pm 3.85$ | $85.15 \pm 7.99$ | $81.25 \pm 8.71$ | $91.2 \pm 6.9$ | $258.0 \pm 24.8$ |
| 1.33 | $4.00 \pm 0.76$ | $18.28 \pm 1.63$ | $84.65 \pm 9.37$ | $87.03 \pm 6.85$ | $247.8 \pm 7.7$ | $374.2 \pm 20.9$ |
| 1.66 | $4.48 \pm 1.63$ | $20.27 \pm 8.84$ | $88.47 \pm 6.68$ | $87.51 \pm 6.28$ | $383.0 \pm 5.7$ | $451.6 \pm 12.0$ |
| 2.00 | $3.31 \pm 0.31$ | $31.52 \pm 5.07$ | $88.94 \pm 6.00$ | $88.95 \pm 6.33$ | $661.2 \pm 10.2$ | $664.8 \pm 35.2$ |
| 2.33 | $3.54 \pm 0.35$ | $51.83 \pm 17.96$ | $88.94 \pm 4.97$ | $88.94 \pm 5.41$ | $770.8 \pm 4.4$ | $763.0 \pm 7.0$ |
| 2.66 | $3.83 \pm 0.38$ | $64.59 \pm 9.19$ | $88.94 \pm 4.97$ | $88.94 \pm 4.97$ | $782.0 \pm 3.4$ | $789.4 \pm 2.8$ |
| 3.00 | $3.96 \pm 0.45$ | $70.08 \pm 9.18$ | $88.94 \pm 4.97$ | $89.92 \pm 5.13$ | $786.0 \pm 4.1$ | $792.2 \pm 1.1$ |

Note that these kernels do not form any special hierarchy so approaches such as [2] are not applicable. Timing results on a log-log scale are given in Figure (1a). As can be seen, SMO-MKL appears to be scaling linearly with the number of kernels and we converge in less than half an hour on all hundred thousand kernels for both $p = 2$ and $p = 1.33$. If we were to run the same experiment using pre-computed kernels then we converge in approximately seven minutes (see Fig (1b)). On the other hand, Shogun took six hundred seconds to combine just ten thousand kernels computed on the fly.

The trend was the same when we increased the number of training points. Figure (1c) and (1d) plot timing results on a log-log scale as the number of training points is varied on the Adult and Web data sets (please see [1] for data set details and downloads). We used 50 kernels computed on the

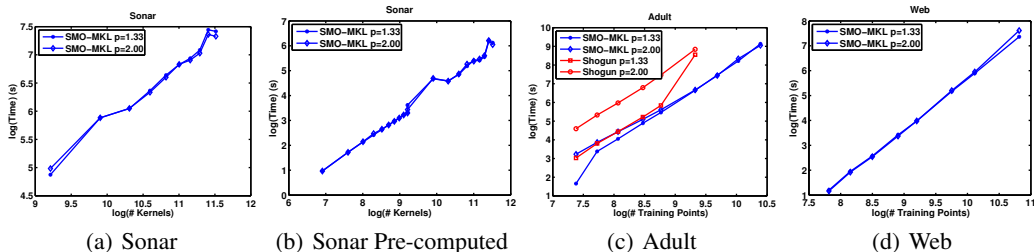

Figure 1: Large scale experiments varying the number of kernels and points. See text for details.

fly for these experiments. On Adult, till about six thousand points, SMO-MKL is roughly 1.5 times faster than Shogun for $p = 1.33$ and 5 times faster for $p = 2$. However, on reaching eleven thousand points, Shogun starts taking more and more time to converge and we could not get results for sixteen thousand points or more. SMO-MKL was unaffected and converged on the full data set with 32,561 points in 9245.80 seconds for $p = 1.33$ and 8511.12 seconds for $p = 2$. We tried the Web data set to see whether the SMO-MKL algorithm would scale beyond 32K points. Training on all 49,749 points and 50 kernels took 1574.73 seconds (i.e. less than half an hour) with $p = 1.33$ and 2023.35 seconds with $p = 2$.

## 7   Conclusions

We developed the SMO-MKL algorithm for efficiently optimising the $l_p$-MKL formulation. We placed the emphasis firmly back on optimising the MKL dual rather than the intermediate saddle point problem on which all state-of-the-art MKL solvers are based. We showed that the $l_p$-MKL dual is differentiable and that placing the $p$-norm squared regulariser in the primal objective lets us analytically solve the reduced variable problem for $p = 2$. We could also solve the convex, one-dimensional reduced variable problem when $p \neq 2$ by the Newton-Raphson method. A second-order working set selection algorithm was implemented to speed up convergence. The resulting algorithm is simple, easy to implement and efficiently scales to large problems. We also showed how to generalise the algorithm to handle not just $p$-norms squared but also certain Bregman divergences.

In terms of empirical performance, we compared the SMO-MKL algorithm to the specialised $l_p$-MKL solver of [12] referred to as Shogun. It was demonstrated that SMO-MKL was significantly faster than Shogun on both small and large scale data sets – sometimes by an order of magnitude. SMO-MKL was also found to be relatively stable for various values of $p$ and could therefore be used to learn both sparse, and non-sparse, kernel combinations. We demonstrated that the algorithm could combine a hundred thousand kernels on Sonar in approximately seven minutes using pre-computed kernels and in less than half an hour using kernels computed on the fly. This is significant as many non-linear kernel combination forms, which lead to performance improvements but are non-convex, can be recast as convex linear MKL with a much larger set of base kernels. The SMO-MKL algorithm can now be used to tackle such problems as long as an appropriate regulariser can be found. We were also able to train on the entire Web data set with nearly fifty thousand points and fifty kernels computed on the fly in less than half an hour. Other solvers were not able to return results on these problems. All experiments were carried out on a single core and therefore, we believe, redefine the state-of-the-art in terms of MKL optimisation. The SMO-MKL code is available for download from [20].

## Acknowledgements

We are grateful to Saurabh Gupta, Marius Kloft and Soren SSonnenburg for helpful discussions, feedback and help with Shogun.

## References

[1] http://www.csie.ntu.edu.tw/ cjlin/libsvmtools/datasets/binary.html.

[2] F. R. Bach. Exploring large feature spaces with hierarchical multiple kernel learning. In *NIPS*, pages 105–112, 2008.

[3] F. R. Bach, G. R. G. Lanckriet, and M. I. Jordan. Multiple kernel learning, conic duality, and the SMO algorithm. In *ICML*, pages 6–13, 2004.

[4] A. Ben-Tal, T. Margalit, and A. Nemirovski. The ordered subsets mirror descent optimization method with applications to tomography. *SIAM Journal of Opimization*, 12(1):79–108, 2001.

[5] C.-C. Chang and C.-J. Lin. *LIBSVM: a library for support vector machines*, 2001. Software available at `http://www.csie.ntu.edu.tw/~cjlin/libsvm`.

[6] C. Cortes, M. Mohri, and A. Rostamizadeh. L2 regularization for learning kernels. In *UAI*, 2009.

[7] C. Cortes, M. Mohri, and A. Rostamizadeh. Learning non-linear combinations of kernels. In *NIPS*, 2009.

[8] R. E. Fan, P. H. Chen, and C. J. Lin. Working set selection using second order information for training SVM. *JMLR*, 6:1889–1918, 2005.

[9] C. Gentile. Robustness of the $p$-norm algorithms. *ML*, 53(3):265–299, 2003.

[10] M. Gonen and E. Alpaydin. Localized multiple kernel learning. In *ICML*, 2008.

[11] J. Kivinen, M. K. Warmuth, and B. Hassibi. The $p$-norm generaliziation of the LMS algorithm for adaptive filtering. *IEEE Trans. Signal Processing*, 54(5):1782–1793, 2006.

[12] M. Kloft, U. Brefeld, S. Sonnenburg, P. Laskov, K.-R. Muller, and A. Zien. Efficient and accurate $l_p$-norm Multiple Kernel Learning. In *NIPS*, 2009.

[13] G. R. G. Lanckriet, N. Cristianini, P. Bartlett, L. El Ghaoui, and M. I. Jordan. Learning the kernel matrix with semidefinite programming. *JMLR*, 5:27–72, 2004.

[14] C. J. Lin, S. Lucidi, L. Palagi, A. Risi, and M. Sciandrone. Decomposition algorithm model for singly linearly-constrained problems subject to lower and upper bounds. *JOTA*, 141(1):107–126, 2009.

[15] J. Platt. Fast training of support vector machines using sequential minimal optimization. In *Advances in Kernel Methods – Support Vector Learning*, pages 185–208, 1999.

[16] A. Rakotomamonjy, F. Bach, Y. Grandvalet, and S. Canu. SimpleMKL. *JMLR*, 9:2491–2521, 2008.

[17] S. Sonnenburg, G. Raetsch, C. Schaefer, and B. Schoelkopf. Large scale multiple kernel learning. *JMLR*, 7:1531–1565, 2006.

[18] M. Varma and B. R. Babu. More generality in efficient multiple kernel learning. In *ICML*, 2009.

[19] A. Vedaldi, V. Gulshan, M. Varma, and A. Zisserman. Multiple kernels for object detection. In *ICCV*, 2009.

[20] S. V. N. Vishwanathan, Z. Sun, N. Theera-Ampornpunt, and M. Varma, 2010. The SMO-MKL code `http://research.microsoft.com/~manik/code/SMO-MKL/download.html`.

[21] J. Yang, Y. Li, Y. Tian, L. Duan, and W. Gao. Group-sensitive multiple kernel learning for object categorization. In *ICCV*, 2009.

